# Improving Topic Coherence with Regularized Topic Models

**David Newman**
University of California, Irvine
newman@uci.edu

**Edwin V. Bonilla**          **Wray Buntine**
NICTA & Australian National University
{edwin.bonilla, wray.buntine}@nicta.com.au

## Abstract

Topic models have the potential to improve search and browsing by extracting useful semantic themes from web pages and other text documents. When learned topics are coherent and interpretable, they can be valuable for faceted browsing, results set diversity analysis, and document retrieval. However, when dealing with small collections or noisy text (e.g. web search result snippets or blog posts), learned topics can be less coherent, less interpretable, and less useful. To overcome this, we propose two methods to *regularize* the learning of topic models. Our regularizers work by creating a structured prior over words that reflect broad patterns in the external data. Using thirteen datasets we show that both regularizers improve topic coherence and interpretability while learning a faithful representation of the collection of interest. Overall, this work makes topic models more useful across a broader range of text data.

## 1   Introduction

Topic modeling holds much promise for improving the ways users search, discover, and organize online content by automatically extracting semantic themes from collections of text documents. Learned topics can be useful in user interfaces for ad-hoc document retrieval [18]; driving faceted browsing [14]; clustering search results [19]; or improving display of search results by increasing result diversity [10]. When the text being modeled is plentiful, clear and well written (e.g. large collections of abstracts from scientific literature), learned topics are usually coherent, easily understood, and fit for use in user interfaces. However, topics are not always consistently coherent, and even with relatively well written text, one can learn topics that are a mix of concepts or hard to understand [1, 6]. This problem is exacerbated for content that is sparse or noisy, such as blog posts, tweets, or web search result snippets. Take for instance the task of learning categories in clustering search engine results. A few searches with Carrot2, Yippee, or WebClust quickly demonstrate that consistently learning meaningful topic facets is a difficult task [5].

Our goal in this paper is to improve the coherence, interpretability and ultimate usability of learned topics. To achieve this we propose QUAD-REG and CONV-REG, two new methods for regularizing topic models, which produce more coherent and interpretable topics. Our work is predicated on recent evidence that a pointwise mutual information-based score (PMI-Score) is highly correlated with human-judged topic coherence [15, 16]. We develop two Bayesian regularization formulations that are designed to improve PMI-Score. We experiment with five search result datasets from 7M Blog posts, four search result datasets from 1M News articles, and four datasets of Google search results. Using these thirteen datasets, our experiments demonstrate that both regularizers consistently improve topic coherence and interpretability, as measured separately by PMI-Score and human judgements. To the best of our knowledge, our models are the first to address the problem of learning topics when dealing with limited and/or noisy text content. This work opens up new application areas for topic modeling.

## 2 Topic Coherence and PMI-Score

Topics learned from a statistical topic model are formally a multinomial distribution over words, and are often displayed by printing the 10 most probable words in the topic. These top-10 words usually provide sufficient information to determine the subject area and interpretation of a topic, and distinguish one topic from another. However, topics learned on sparse or noisy text data are often less coherent, difficult to interpret, and not particularly useful. Some of these *noisy* topics can be vaguely interpretable, but contain (in the top-10 words) one or two unrelated words – while other topics can be practically incoherent. In this paper we wish to improve topic models learned on document collections where the text data is sparse and/or noisy. We postulate that using additional (possibly external) data will regularize the learning of the topic models.

Therefore, our goal is to **improve topic coherence**. Topic coherence – meaning semantic coherence – is a human judged quality that depends on the semantics of the words, and cannot be measured by model-based statistical measures that treat the words as exchangeable tokens. Fortunately, recent work has demonstrated that it is possible to automatically measure topic coherence with near-human accuracy [16, 15] using a score based on pointwise mutual information (PMI). In that work they showed (using 6000 human evaluations) that the PMI-Score broadly agrees with human-judged topic coherence. The PMI-Score is motivated by measuring word association between all pairs of words in the top-10 topic words. PMI-Score is defined as follows:

$$\text{PMI-Score}(\mathbf{w}) = \frac{1}{45} \sum_{i<j} \text{PMI}(w_i, w_j), ij \in \{1 \dots 10\} \tag{1}$$

$$\text{where} \quad \text{PMI}(w_i, w_j) = \log \frac{P(w_i, w_j)}{P(w_i)P(w_j)}, \tag{2}$$

and 45 is the number of PMI scores over the set of distinct word pairs in the top-10 words. A key aspect of this score is that it uses *external* data – that is data not used during topic modeling. This data could come from a variety of sources, for example the corpus of 3M English Wikipedia articles.

For this paper, we will use both PMI-Score and human judgements to measure topic coherence. Note that we can measure the PMI-Score of an individual topic, or for a topic model of $T$ topics (in that case PMI-Score will refer to the average of $T$ PMI-Scores). This PMI-Score – and the idea of using external data to measure it – forms the foundation of our idea for regularization.

## 3 Regularized Topic Models

In this section we describe our approach to regularization in topic models by proposing two different methods: (a) a quadratic regularizer (QUAD-REG) and (b) a convolved Dirichlet regularizer (CONV-REG). We start by introducing the standard notation in topic modeling and the baseline latent Dirichlet allocation method (LDA, [4, 9]).

### 3.1 Topic Modeling and LDA

Topic models are a Bayesian version of probabilistic latent semantic analysis [11]. In standard LDA topic modeling each of $D$ documents in the corpus is modeled as a discrete distribution over $T$ latent topics, and each topic is a discrete distribution over the vocabulary of $W$ words. For document $d$, the distribution over topics, $\theta_{t|d}$, is drawn from a Dirichlet distribution Dir$[\alpha]$. Likewise, each distribution over words, $\phi_{w|t}$, is drawn from a Dirichlet distribution, Dir$[\beta]$.

For the $i^{th}$ token in a document, a topic assignment, $z_{id}$, is drawn from $\theta_{t|d}$ and the word, $x_{id}$, is drawn from the corresponding topic, $\phi_{w|z_{id}}$. Hence, the generative process in LDA is given by:

$$\theta_{t|d} \sim \text{Dirichlet}[\alpha] \qquad \phi_{w|t} \sim \text{Dirichlet}[\beta] \tag{3}$$

$$z_{id} \sim \text{Mult}[\theta_{t|d}] \qquad x_{id} \sim \text{Mult}[\phi_{w|z_{id}}]. \tag{4}$$

We can compute the posterior distribution of the topic assignments via Gibbs sampling by writing down the joint probability, integrating out $\theta$ and $\phi$, and following a few simple mathematical manipulations to obtain the standard Gibbs sampling update:

$$p(z_{id} = t | x_{id} = w, \mathbf{z}^{\neg i}) \propto \frac{N_{wt}^{\neg i} + \beta}{N_t^{\neg i} + W\beta}(N_{td}^{\neg i} + \alpha). \tag{5}$$

where $\mathbf{z}^{\neg i}$ denotes the set of topic assignment variables except the $i^{th}$ variable; $N_{wt}$ is the number of times word $w$ has been assigned to topic $t$; $N_{td}$ is the number of times topic $t$ has been assigned to document $d$, and $N_t = \sum_{w=1}^{W} N_{wt}$.

Given samples from the posterior distribution we can compute point estimates of the document-topic proportions $\theta_{t|d}$ and the word-topic probabilities $\phi_{w|t}$. We will denote henceforth $\boldsymbol{\phi}_t$ as the vector of word probabilities for a given topic $t$ and analogously for other variables.

## 3.2 Regularization via Structured Priors

To learn better topic models for small or noisy collections we introduce structured priors on $\boldsymbol{\phi}_t$ based upon external data, which has a regularization effect on the standard LDA model. More specifically, our priors on $\boldsymbol{\phi}_t$ will depend on the structural relations of the words in the vocabulary as given by external data, which will be characterized by the $W \times W$ "covariance" matrix $\mathbf{C}$. Intuitively, $\mathbf{C}$ is a matrix that captures the short-range dependencies between words in the external data. More importantly, we are only interested in relatively frequent terms from the vocabulary, so $\mathbf{C}$ will be a sparse matrix and hence computations are still feasible for our methods to be used in practice.

## 3.3 Quadratic Regularizer (QUAD-REG)

Here we use a standard quadratic form with a trade-off factor. Therefore, given a matrix of word dependencies $\mathbf{C}$, we can use the prior:

$$p(\boldsymbol{\phi}_t|\mathbf{C}) \propto \left( \boldsymbol{\phi}_t^T \mathbf{C} \boldsymbol{\phi}_t \right)^\nu \tag{6}$$

for some power $\nu$. Note we do not know the normalization factor but for our purposes of MAP estimation we do not need it. The log posterior (omitting irrelevant constants) is given by:

$$\mathcal{L}_{\text{MAP}} = \sum_{i=1}^{W} N_{it} \log \phi_{i|t} + \nu \log \left( \boldsymbol{\phi}_t^T \mathbf{C} \boldsymbol{\phi}_t \right). \tag{7}$$

Optimizing Equation (7) with respect to $\phi_{w|t}$ subject to the constraints $\sum_{i=1}^{W} \phi_{i|t} = 1$, we obtain the following fixed point update:

$$\phi_{w|t} \leftarrow \frac{1}{N_t + 2\nu} \left( N_{wt} + 2\nu \frac{\phi_{w|t} \sum_{i=1}^{W} \mathrm{C}_{iw} \phi_{i|t}}{\boldsymbol{\phi}_t^T \mathbf{C} \boldsymbol{\phi}_t} \right). \tag{8}$$

We note that unlike other topic models in which a covariance or correlation structure is used (as in the correlated topic model, [3]) in the context of correlated priors for $\theta_{t|d}$, our method does not require the inversion of $\mathbf{C}$, which would be impractical for even modest vocabulary sizes.

By using the update in Equation (8) we obtain the values for $\phi_{w|t}$. This means we no longer have neat conjugate priors for $\phi_{w|t}$ and thus the sampling in Equation (5) does not hold. Instead, at the end of each major Gibbs cycle, $\phi_{w|t}$ is re-estimated and the corresponding Gibbs update becomes:

$$p(z_{id} = t | x_{id} = w, \mathbf{z}^{\neg i}, \phi_{w|t}) \propto \phi_{w|t} (N_{td}^{\neg i} + \alpha). \tag{9}$$

## 3.4 Convolved Dirichlet Regularizer (CONV-REG)

Another approach to leveraging information on word dependencies from external data is to consider that each $\boldsymbol{\phi}_t$ is a mixture of word probabilities $\boldsymbol{\psi}_t$, where the coefficients are constrained by the word-pair dependency matrix $\mathbf{C}$:

$$\boldsymbol{\phi}_t \propto \mathbf{C} \boldsymbol{\psi}_t \qquad \text{where} \quad \boldsymbol{\psi}_t \sim \text{Dirichlet}(\gamma \mathbf{1}). \tag{10}$$

Each topic has a different $\boldsymbol{\psi}_t$ drawn from a Dirichlet, thus the model is a convolved Dirichlet. This means that we convolve the supplied topic to include a spread of related words. Then we have that:

$$p(\mathbf{w}|z = t, \mathbf{C}, \psi_t) = \prod_{i=1}^{W} \left( \sum_{j=1}^{W} \mathrm{C}_{ij} \psi_{j|t} \right)^{N_{it}}. \tag{11}$$

Table 1: Search result datasets came from a collection of 7M Blogs, a collection of 1M News articles, and the web. The first two collections were indexed with Lucene. The queries below were issued to create five Blog datasets, four News datasets, and four Web datasets. Search result set sizes ranged from 1000 to 18,590. For Blogs and News, half of each dataset was set aside for Test, and Train was sampled from the remaining half. For Web, Train was the top-40 search results.

| | Name | Query | # Results | $D_{\text{Test}}$ | $D_{\text{Train}}$ |
|---|---|---|---|---|---|
| Blogs | beijing | beijing olympic ceremony | 5024 | 2512 | 39 |
| | climate | climate change | 14,932 | 7466 | 58 |
| | obama | obama debate | 18,590 | 9295 | 72 |
| | palin | palin interview | 10,478 | 5239 | 40 |
| | vista | vista problem | 4214 | 2107 | 32 |
| News | baseball | major league baseball game team player | 3774 | 1887 | 29 |
| | drama | television show series drama | 3024 | 1512 | 23 |
| | health | health medicine insurance | 1655 | 828 | 25 |
| | legal | law legal crime court | 2976 | 1488 | 23 |
| Web | depression | depression | 1000 | 1000 | 40 |
| | migraine | migraine | 1000 | 1000 | 40 |
| | america | america | 1000 | 1000 | 40 |
| | south africa | south africa | 1000 | 1000 | 40 |

We obtain the MAP solution to $\psi_t$ by optimizing:

$$\mathcal{L}_{\text{MAP}} = \sum_{i=1}^{W} N_{it} \log \sum_{j=1}^{W} \mathrm{C}_{ij} \psi_{j|t} + \sum_{j=1}^{W} (\gamma - 1) \log \psi_{j|t} \qquad \text{s.t.} \sum_{j=1}^{W} \psi_{j|t} = 1. \qquad (12)$$

Solving for $\psi_{w|t}$ we obtain:

$$\psi_{w|t} \propto \sum_{i=1}^{W} \frac{N_{it} \mathrm{C}_{iw}}{\sum_{j=1}^{W} \mathrm{C}_{ij} \psi_{j|t}} \psi_{w|t} + \gamma. \qquad (13)$$

We follow the same semi-collapsed inference procedure used for QUAD-REG, with the updates in Equations (13) and (10) producing the values for $\phi_{w|t}$ to be used in the semi-collapsed sampler (9).

## 4   Search Result Datasets

Text datasets came from a collection of 7M Blogs (from ICWSM 2009), a collection of 1M News articles (LDC Gigaword), and the Web (using Google's search). Table 1 shows a summary of the datasets used. These datasets provide a diverse range of content for topic modeling. Blogs are often written in a chatty and informal style, which tends to produce topics that are difficult to interpret. News articles are edited to a higher standard, so learned topics are often fairly interpretable when one models, say, thousands of articles. However, our experiments use 23-29 articles, limiting the data for topic learning. Snippets from web search result present perhaps the most sparse data. For each dataset we created the standard bag-of-words representation and performed fairly standard tokenization. We created a vocabulary of terms that occurred at least five times (or two times, for the Web datasets), after excluding stopwords. We learned the topic models on the *Train* data set, setting $T = 15$ for Blogs datasets, $T = 10$ for News datasets, and $T = 8$ for the Web datasets.

**Construction of C**: The word co-occurrence data for regularization was obtained from the entire LDC dataset of 1M articles (for News), a subset of the 7M blog posts (for Blogs), and using all 3M English Wikipedia articles (for Web). Word co-occurrence was computed using a sliding window of ten words to emphasize short-range dependency. Note that we only kept positive PMI values. For each dataset we created a $W \times W$ matrix of co-occurrence counts using the 2000-most frequent terms in the vocabulary for that dataset, thereby maintaining reasonably good sparsity for these data. Selecting most-frequent terms makes sense because our objective is to improve PMI-Score, which is defined over the top-10 topic words, which tend to involve relatively high-frequency terms. Using high-frequency terms also avoids potential numerical problems of large PMI values arising from co-occurrence of rare terms.

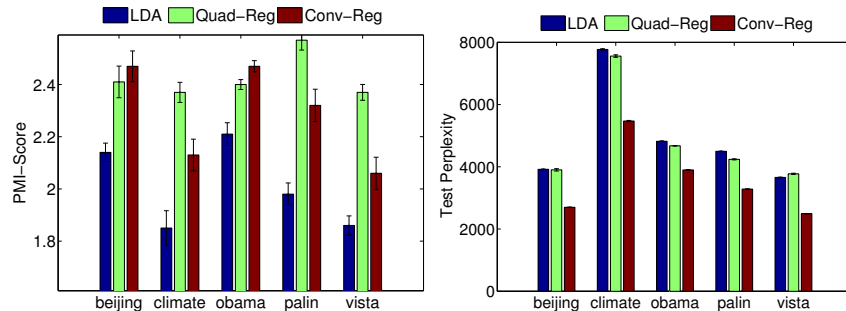

Figure 1: PMI-Score and test perplexity of regularized methods vs. LDA on Blogs, $T = 15$. Both regularization methods improve PMI-Score and perplexity for all datasets, with the exception of 'vista' where QUAD-REG has slightly higher perplexity.

## 5 Experiments

In this section we evaluate our regularized topic models by reporting the average PMI-Score over 10 different runs, each computed using Equations (1) and (2) (and then in Section 5.4, we use human judgements). Additionally, we report the average test data perplexity over 10 samples from the posterior across ten independent chains, where each perplexity is calculated using:

$$\text{Perp}(x^{\text{test}}) = \exp\left(-\frac{1}{N^{\text{test}}}\log p(x^{\text{test}})\right) \qquad \log p(x^{\text{test}}) = \sum_{dw} N_{dw}^{\text{test}} \log \sum_t \phi_{w|t}\theta_{t|d} \qquad (14)$$

$$\theta_{t|d} = \frac{\alpha + N_{td}}{T\alpha + N_d} \qquad\qquad \phi_{w|t} = \frac{\beta + N_{wt}}{W\beta + N_t}. \qquad (15)$$

The document mixture $\theta_{t|d}$ is learned from test data, and the log probability of the test words is computed using this mixture. Each $\phi_{w|t}$ is computed by Equation (15) for the baseline LDA model, and it is used directly for the QUAD-REG and CONV-REG methods. For the Gibbs sampling algorithms we set $\alpha = 0.05\frac{N}{DT}$ and $\beta = 0.01$ (initially). This setting of $\alpha$ allocates $5\%$ of the probability mass for smoothing. We run the sampling for 300 iterations; applied the fixed point iterations (on the regularized models) 10 times every 20 Gibbs iterations and ran 10 different random initializations (computing average over these runs). We used $T = 10$ for the News datasets, $T = 15$ for the Blogs datasets and $T = 8$ for the Web datasets. Note that test perplexity is computed on $D_{\text{Test}}$ (Table 1) that is at least an order of magnitude larger than the training data. After some preliminary experiments, we fixed QUAD-REG's regularization parameter to $\nu = 0.5\frac{N}{T}$.

### 5.1 Results

Figures 1 and 2 show the average PMI-Scores and average test perplexities for the Blogs and News datasets. For Blogs (Figure 1) we see that our regularized models consistently improve PMI-Score and test perplexity on all datasets with the exception of the 'vista' dataset where QUAD-REG has slightly higher perplexity. For News (Figure 2) we see that both regularization methods improve PMI-Score and perplexity for all datasets. Hence, we can conclude that our regularized models not only provide a good characterization of the collections but also improve the coherence of the learned topics as measured by the PMI-Score. It is reasonable to expect both PMI-Score and perplexity to improve as semantically related words should be expected in topic models, so with little data, our regularizers push both measures in a positive direction.

### 5.2 Coherence of Learned Topics

Table 2 shows selected topics learned by LDA and our QUAD-REG model. To obtain correspondence of topics (for this experiment), we initialized the QUAD-REG model with the converged LDA model. Overall, our regularized model tends to learn topics that are more focused on a particular subject, contain fewer spurious words, and therefore are easier to interpret. The following list explains how the regularized version of the topic is more useful:

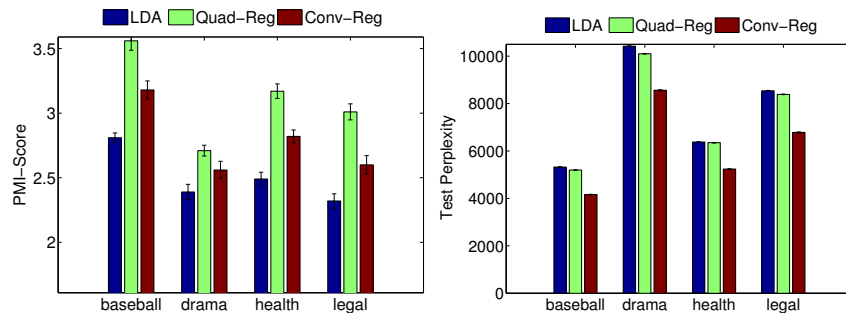

Figure 2: PMI-Score and test perplexity of regularized methods vs. LDA on News, $T = 10$. Both regularization methods improve PMI-Score and perplexity for all datasets.

Table 2: Selected topics improved by regularization. Each pair first shows an LDA topic and the corresponding topic produced by QUAD-REG (initialized from the converged LDA model). QUAD-REG's PMI-Scores were always better than LDA's on these examples. The regularized versions tend to be more focused on a particular subject and easier to interpret.

| Name | Model | Topic |
|---|---|---|
| beijing | LDA | girl phony world yang fireworks interest maybe miaoke peiyi young |
|  | REG | girl yang peiyi miaoke lin voice real lip music sync |
| obama | LDA | palin biden sarah running mccain media hilton stein paris john |
|  | REG | palin sarah mate running biden vice governor selection alaska choice |
| drama | LDA | wire david place police robert baltimore corner friends com simon |
|  | REG | drama episode characters series cop cast character actors detective emmy |
| legal | LDA | saddam american iraqi iraq judge against charges minister thursday told |
|  | REG | iraqi saddam iraq military crimes tribunal against troops accused officials |

**beijing** QUAD-REG has better focus on the names and issues involved in the controversy over the Chinese replacing the young girl doing the actual singing at the Olympic opening ceremony with the girl who lip-synched.

**obama** QUAD-REG focuses on Sarah Palin's selection as a GOP Vice Presidential candidate, while LDA has a less clear theme including the story of Paris Hilton giving Palin fashion advice.

**drama** QUAD-REG learns a topic related to television police dramas, while LDA narrowly focuses on David Simon's *The Wire* along with other scattered terms: *robert* and *friends*.

**legal** LDA topic is somewhat related to Saddam Hussein's appearance in court, but includes uninteresting terms such as: *thursday*, and *told*. The QUAD-REG topic is an overall better category relating to the tribunal and charges against Saddam Hussein.

## 5.3 Modeling of Google Search Results

Are our regularized topic models useful for building facets in a clustering-web-search-results type of application? Figure 3 (top) shows the average PMI-Score (mean $+/-$ two standard errors over 10 runs) for the four searches described in Table 1 (Web dataset) and the average perplexity using top-1000 results as test data (bottom). In all cases QUAD-REG and CONV-REG learn better topics, as measured by PMI-Score, compared to those learned by LDA. Additionally, whereas QUAD-REG exhibits slightly higher values of perplexity compared to LDA, CONV-REG consistently improved perplexity on all four search datasets. This level of improvement in PMI-Score through regularization was not seen in News or Blogs likely because of the greater sparsity in these data.

## 5.4 Human Evaluation of Regularized Topic Models

So far we have evaluated our regularized topic models by assessing (a) how faithful their representation is to the collection of interest, as measured by test perplexity, and (b) how coherent they are,

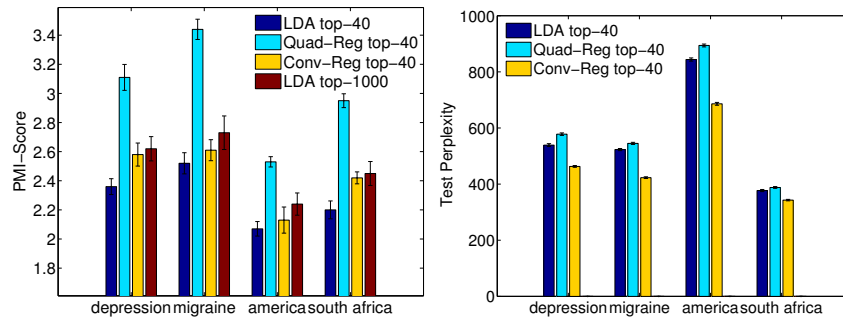

Figure 3: PMI-Score and test perplexity of regularized methods vs. LDA on Google search results. Both methods improve PMI-Score and CONV-REG also improves test perplexity, which is computed using top-1000 results as test data (therefore top-1000 test perplexity is not reported).

as given by the PMI-Score. Ultimately, we have hypothesized that humans will find our regularized topic models more semantically coherent than baseline LDA and therefore more useful for tasks such as document clustering, search and browsing. To test this hypothesis we performed further experiments where we asked humans to directly compare our regularized topics with LDA topics and choose which is more coherent. As our experimental results in this section show, our regularized topic models significantly outperform LDA based on actual human judgements.

To evaluate our models with human judgments we used Amazon Mechanical Turk (AMT, `https://www.mturk.com`) where we asked workers to compare topic pairs (one topic given by one of our regularized models and the other topic given by LDA) and to answer explicitly which topic was more coherent according to how clearly they represented a single theme/idea/concept. To keep the cognitive load low (while still having a fair and sound evaluation of the topics) we described each topic by its top-10 words. We provided an additional option "...*Can't decide*..." indicating that the user could not find a qualitative difference between the topics presented. We also included *control* comparisons to filter out bad workers. These control comparisons were done by replacing a randomly-selected topic word with an intruder word. To have aligned (matched) pairs of topics, the sampling procedure of our regularized topic models was initialized with LDA's topic assignment obtained after convergence of Gibbs sampling. These experiments produced a total of 3650 topic-comparison human evaluations and the results can be seen in Figure 4.

## 6 Related Work

Several authors have investigated the use of domain knowledge from external sources in topic modeling. For example, [7, 8] propose a method for combining topic models with ontological knowledge to tag web pages. They constrain the topics in an LDA-based model to be amongst those in the given ontology. [20] also use statistical topic models with a predefined set of topics to address the task of query classification. Our goal is different to theirs in that we are not interested in constraining the learned topics to those in the external data but rather in improving the topics in small or noisy collections by means of regularization. Along a similar vein, [2] incorporate domain knowledge into topic models by encouraging some word pairs to have similar probability within a topic. Their method, as ours, is based on replacing the standard Dirichlet prior over word-topic probabilities. However, unlike our approach that is entirely data-driven, it appears that their method relies on interactive feedback from the user or on the careful selection of words within an ontological concept.

The effect of structured priors in LDA has been investigated by [17] who showed that learning hierarchical Dirichlet priors over the document-topic distribution can provide better performance than using a symmetric prior. Our work is motivated by the fact that priors matter but is focused on a rather different use case of topic models, i.e. when we are dealing with small or noisy collections and want to improve the coherence of the topics by re-defining the prior on the word-topic distributions.

Priors that introduce correlations in topic models have been investigated by [3]. Unlike our work that considers priors on the word-topic distributions ($\phi_{w|t}$), they introduce a correlated prior on the

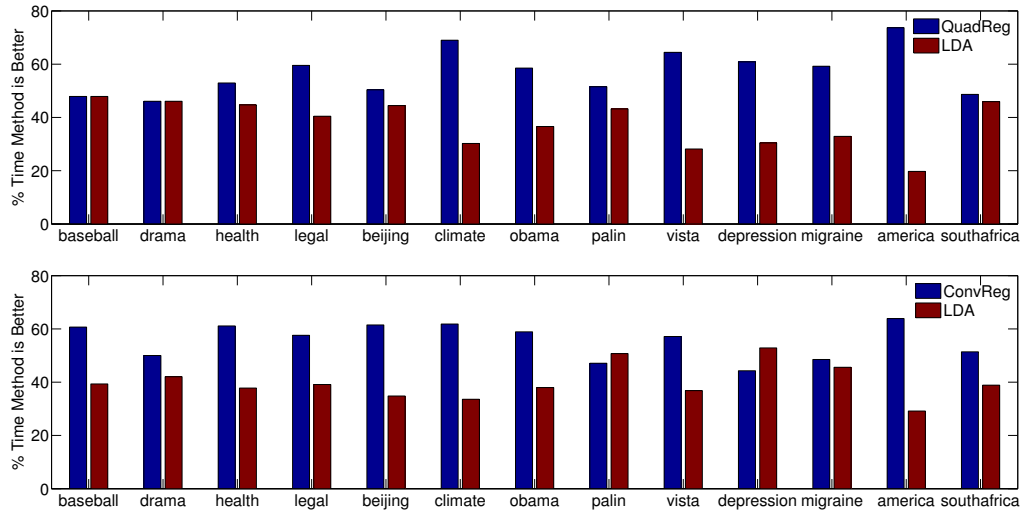

Figure 4: The proportion of times workers in Amazon Mechanical Turk selected each topic model as showing better coherence. In nearly all cases our regularized models outperform LDA. CONV-REG outperforms LDA in 11 of 13 datasets. QUAD-REG never performs worse than LDA (at the dataset level). On average (from 3650 topic comparisons) workers selected QUAD-REG as more coherent 57% of the time while they selected LDA as more coherent only 37% of the time. Similarly, they chose CONV-REG's topics as more coherent 56% of the time, and LDA as more coherent only 39% of the time. These results are statistically significant at 5% level of significance when performing a paired t-test on the total values across all datasets. Note that the two bars corresponding to each dataset do not add up to 100% as the remaining mass corresponds to "...*Can't decide*..." responses.

topic proportions ($\theta_{t|d}$). In our approach, considering similar priors for $\phi_{w|t}$ to those studied by [3] would be unfeasible as they would require the inverse of a $W \times W$ covariance matrix.

Network structures associated with a collection of documents are used in [12] in order to "smooth" the topic distributions of the PLSA model [11]. Our methods are different in that they do not require the collection under study to have an associated network structure as we aim at addressing the different problem of regularizing topic models on small or noisy collections. Additionally, their work is focused on regularizing the document-topic distributions instead of the word-topic distributions. Finally, the work in [13], contemporary to ours, also addresses the problem of improving the quality of topic models. However, our approach focuses on exploiting the knowledge provided by external data given the noisy and/or small nature of the collection of interest.

# 7    Discussion & Conclusions

In this paper we have proposed two methods for regularization of LDA topic models based upon the direct inclusion of word dependencies in our word-topic prior distributions. We have shown that our regularized models can improve the coherence of learned topics significantly compared to the baseline LDA method, as measured by the PMI-Score and assessed by human workers in Amazon Mechanical Turk. While our focus in this paper has been on small, and small *and* noisy datasets, we would expect our regularization methods also to be effective on large and noisy datasets. Note that mixing and rate of convergence may be more of an issue with larger datasets, since our regularizers use a semi-collapsed Gibbs sampler. We will address these large noisy collections in future work.

**Acknowledgments**

NICTA is funded by the Australian Government as represented by the Department of Broadband, Communications and the Digital Economy and the Australian Research Council through the ICT Centre of Excellence program. DN was also supported by an NSF EAGER Award, an IMLS Research Grant, and a Google Research Award.

# References

[1] L. AlSumait, D. Barbará, J. Gentle, and C. Domeniconi. Topic significance ranking of LDA generative models. In *ECML/PKDD*, 2009.

[2] D. Andrzejewski, X. Zhu, and M. Craven. Incorporating domain knowledge into topic modeling via Dirichlet forest priors. In *ICML*, 2009.

[3] David M. Blei and John D. Lafferty. Correlated topic models. In *NIPS*, 2005.

[4] D.M. Blei, A.Y. Ng, and M.I. Jordan. Latent Dirichlet allocation. *JMLR*, 3:993–1022, 2003.

[5] Claudio Carpineto, Stanislaw Osinski, Giovanni Romano, and Dawid Weiss. A survey of web clustering engines. *ACM Comput. Surv.*, 41(3), 2009.

[6] J. Chang, J. Boyd-Graber, S. Gerrish, C. Wang, and D. Blei. Reading tea leaves: How humans interpret topic models. In *NIPS*, 2009.

[7] Chaitanya Chemudugunta, America Holloway, Padhraic Smyth, and Mark Steyvers. Modeling documents by combining semantic concepts with unsupervised statistical learning. In *ISWC*, 2008.

[8] Chaitanya Chemudugunta, Padhraic Smyth, and Mark Steyvers. Combining concept hierarchies and statistical topic models. In *CIKM*, 2008.

[9] T. Griffiths and M. Steyvers. Probabilistic topic models. In *Latent Semantic Analysis: A Road to Meaning*, 2006.

[10] Shengbo Guo and Scott Sanner. Probabilistic latent maximal marginal relevance. In *SIGIR*, 2010.

[11] Thomas Hofmann. Probabilistic latent semantic indexing. In *SIGIR*, 1999.

[12] Qiaozhu Mei, Deng Cai, Duo Zhang, and ChengXiang Zhai. Topic modeling with network regularization. In *WWW*, 2008.

[13] David Mimno, Hanna Wallach, Edmund Talley, Miriam Leenders, and Andrew McCallum. Optimizing semantic coherence in topic models. In *EMNLP*, 2011.

[14] D.M. Mimno and A. McCallum. Organizing the OCA: learning faceted subjects from a library of digital books. In *JCDL*, 2007.

[15] D. Newman, J.H. Lau, K. Grieser, and T. Baldwin. Automatic evaluation of topic coherence. In *NAACL HLT*, 2010.

[16] D. Newman, Y. Noh, E. Talley, S. Karimi, and T. Baldwin. Evaluating topic models for digital libraries. In *JCDL*, 2010.

[17] H. Wallach, D. Mimno, and A. McCallum. Rethinking LDA: Why priors matter. In *NIPS*, 2009.

[18] Xing Wei and W. Bruce Croft. LDA-based document models for ad-hoc retrieval. In *SIGIR*, 2006.

[19] Hua-Jun Zeng, Qi-Cai He, Zheng Chen, Wei-Ying Ma, and Jinwen Ma. Learning to cluster web search results. In *SIGIR*, 2004.

[20] Haijun Zhai, Jiafeng Guo, Qiong Wu, Xueqi Cheng, Huawei Sheng, and Jin Zhang. Query classification based on regularized correlated topic model. In *Proceedings of the International Joint Conference on Web Intelligence and Intelligent Agent Technology*, 2009.

